# Principles of Risk Minimization
# for Learning Theory

**V. Vapnik**
AT&T Bell Laboratories
Holmdel, NJ 07733, USA

## Abstract

Learning is posed as a problem of function estimation, for which two princi-
ples of solution are considered: empirical risk minimization and structural
risk minimization. These two principles are applied to two different state-
ments of the function estimation problem: global and local. Systematic
improvements in prediction power are illustrated in application to zip-code
recognition.

## 1 INTRODUCTION

The structure of the theory of learning differs from that of most other theories for
applied problems. The search for a solution to an applied problem usually requires
the three following steps:

1. State the problem in mathematical terms.
2. Formulate a general principle to look for a solution to the problem.
3. Develop an algorithm based on such general principle.

The first two steps of this procedure offer in general no major difficulties; the
third step requires most efforts, in developing computational algorithms to solve
the problem at hand.

In the case of learning theory, however, many algorithms have been developed, but
we still lack a clear understanding of the mathematical statement needed to describe
the learning procedure, and of the general principle on which the search for solutions

should be based. This paper is devoted to these first two steps, the statement of the problem and the general principle of solution.

The paper is organized as follows. First, the problem of function estimation is stated, and two principles of solution are discussed: the principle of empirical risk minimization and the principle of structural risk minimization. A new statement is then given: that of local estimation of function, to which the same principles are applied. An application to zip-code recognition is used to illustrate these ideas.

## 2    FUNCTION ESTIMATION MODEL

The learning process is described through three components:

1. A generator of random vectors $x$, drawn independently from a fixed but unknown distribution $P(x)$.
2. A supervisor which returns an output vector $y$ to every input vector $x$, according to a conditional distribution function $P(y|x)$, also fixed but unknown.
3. A learning machine capable of implementing a set of functions $f(x, w)$, $w \in W$.

The problem of learning is that of choosing from the given set of functions the one which approximates best the supervisor's response. The selection is based on a training set of $\ell$ independent observations:

$$(x_1, y_1), ..., (x_\ell, y_\ell). \tag{1}$$

The formulation given above implies that learning corresponds to the problem of function approximation.

## 3    PROBLEM OF RISK MINIMIZATION

In order to choose the best available approximation to the supervisor's response, we measure the loss or discrepancy $L(y, f(x, w))$ between the response $y$ of the supervisor to a given input $x$ and the response $f(x, w)$ provided by the learning machine. Consider the expected value of the loss, given by the risk functional

$$R(w) = \int L(y, f(x, w))dP(x, y). \tag{2}$$

The goal is to minimize the risk functional $R(w)$ over the class of functions $f(x, w)$, $w \in W$. But the joint probability distribution $P(x, y) = P(y|x)P(x)$ is unknown and the only available information is contained in the training set (1).

## 4    EMPIRICAL RISK MINIMIZATION

In order to solve this problem, the following induction principle is proposed: the risk functional $R(w)$ is replaced by the empirical risk functional

$$E(w) = \frac{1}{\ell} \sum_{i=1}^{\ell} L(y_i, f(x_i, w)) \tag{3}$$

constructed on the basis of the training set (1). The induction principle of empirical risk minimization (ERM) assumes that the function $f(x, w_\ell^*)$, which minimizes $E(w)$ over the set $w \in W$, results in a risk $R(w_\ell^*)$ which is close to its minimum.

This induction principle is quite general; many classical methods such as least square or maximum likelihood are realizations of the ERM principle.

The evaluation of the soundness of the ERM principle requires answers to the following two questions:

1. Is the principle consistent? (Does $R(w_\ell^*)$ converge to its minimum value on the set $w \in W$ when $\ell \to \infty$?)

2. How fast is the convergence as $\ell$ increases?

The answers to these two questions have been shown (Vapnik et al., 1989) to be equivalent to the answers to the following two questions:

1. Does the empirical risk $E(w)$ converge uniformly to the actual risk $R(w)$ over the full set $f(x, w)$, $w \in W$? Uniform convergence is defined as

$$\text{Prob}\{ \sup_{w \in W} |R(w) - E(w)| > \varepsilon \} \longrightarrow 0 \quad \text{as} \quad \ell \to \infty. \tag{4}$$

2. What is the rate of convergence?

It is important to stress that uniform convergence (4) for the full set of functions is a *necessary* and *sufficient* condition for the consistency of the ERM principle.

## 5    VC-DIMENSION OF THE SET OF FUNCTIONS

The theory of uniform convergence of empirical risk to actual risk developed in the 70's and 80's, includes a description of necessary and sufficient conditions as well as bounds for the rate of convergence (Vapnik, 1982). These bounds, which are independent of the distribution function $P(x, y)$, are based on a quantitative measure of the capacity of the set of functions implemented by the learning machine: the VC-dimension of the set.

For simplicity, these bounds will be discussed here only for the case of binary pattern recognition, for which $y \in \{0, 1\}$ and $f(x, w)$, $w \in W$ is the class of indicator functions. The loss function takes only two values $L(y, f(x, w)) = 0$ if $y = f(x, w)$ and $L(y, f(x, w)) = 1$ otherwise. In this case , the risk functional (2) is the probability of error, denoted by $P(w)$. The empirical risk functional (3), denoted by $\nu(w)$, is the frequency of error in the training set.

The VC-dimension of a set of indicator functions is the maximum number $h$ of vectors which can be shattered in all possible $2^h$ ways using functions in the set. For instance, $h = n + 1$ for linear decision rules in $n$-dimensional space, since they can shatter at most $n + 1$ points.

## 6    RATES OF UNIFORM CONVERGENCE

The notion of VC-dimension provides a bound to the rate of uniform convergence. For a set of indicator functions with VC-dimension $h$, the following inequality holds:

$$\text{Prob}\{\sup_{w \in W} |P(w) - \nu(w)| > \varepsilon\} < (\frac{2\ell e}{h})^h \exp\{-\varepsilon^2 \ell\}. \qquad (5)$$

It then follows that with probability $1 - \eta$, simultaneously for all $w \in W$,

$$P(w) < \nu(w) + C_0(\ell/h, \eta), \qquad (6)$$

with confidence interval

$$C_0(\ell/h, \eta) = \sqrt{\frac{h(\ln 2\ell/h + 1) - \ln \eta}{\ell}}. \qquad (7)$$

This important result provides a bound to the actual risk $P(w)$ for all $w \in W$, including the $w^*$ which minimizes the empirical risk $\nu(w)$.

The deviation $|P(w) - \nu(w)|$ in (5) is expected to be maximum for $P(w)$ close to $1/2$, since it is this value of $P(w)$ which maximizes the error variance $\sigma(w) = \sqrt{P(w)(1 - P(w))}$. The worst case bound for the confidence interval (7) is thus likely be controlled by the worst decision rule. The bound (6) is achieved for the worst case $P(w) = 1/2$, but not for small $P(w)$, which is the case of interest. A uniformly good approximation to $P(w)$ follows from considering

$$\text{Prob}\{\sup_{w \in W} \frac{P(w) - \nu(w)}{\sigma(w)} > \varepsilon\}. \qquad (8)$$

The variance of the relative deviation $(P(w) - \nu(w))/\sigma(w)$ is now independent of $w$. A bound for the probability (8), if available, would yield a uniformly good bound for actual risks for all $P(w)$.

Such a bound has not yet been established. But for $P(w) << 1$, the approximation $\sigma(w) \simeq \sqrt{P(w)}$ is true, and the following inequality holds:

$$\text{Prob}\{\sup_{w \in W} \frac{P(w) - \nu(w)}{\sqrt{P(w)}} > \varepsilon\} < (\frac{2\ell e}{h})^h \exp\{-\frac{\varepsilon^2 \ell}{4}\}. \qquad (9)$$

It then follows that with probability $1 - \eta$, simultaneously for all $w \in W$,

$$P(w) < \nu(w) + C_1(\ell/h, \nu(w), \eta), \qquad (10)$$

with confidence interval

$$C_1(\ell/h, \nu(w), \eta) = 2 \left( \frac{h(\ln 2\ell/h + 1) - \ln \eta}{\ell} \right) \left( 1 + \sqrt{1 + \frac{\nu(w)\ell}{h(\ln 2\ell/h + 1) - \ln \eta}} \right). \qquad (11)$$

Note that the confidence interval now depends on $\nu(w)$, and that for $\nu(w) = 0$ it reduces to

$$C_1(\ell/h, 0, \eta) = 2C_0^2(\ell/h, \eta),$$

which provides a more precise bound for real case learning.

# 7    STRUCTURAL RISK MINIMIZATION

The method of ERM can be theoretically justified by considering the inequalities (6) or (10). When $\ell/h$ is large, the confidence intervals $C_0$ or $C_1$ become small, and

can be neglected. The actual risk is then bound by only the empirical risk, and the probability of error on the test set can be expected to be small when the frequency of error in the training set is small.

However, if $\ell/h$ is small, the confidence interval cannot be neglected, and even $\nu(w) = 0$ does not guarantee a small probability of error. In this case the minimization of $P(w)$ requires a new principle, based on the simultaneous minimization of $\nu(w)$ and the confidence interval. It is then necessary to control the VC-dimension of the learning machine.

To do this, we introduce a nested structure of subsets $S_p = \{f(x, w), w \in W_p\}$, such that

$$S_1 \subset S_2 \subset ... \subset S_n.$$

The corresponding VC-dimensions of the subsets satisfy

$$h_1 < h_2 < ... < h_n.$$

The principle of structure risk minimization (SRM) requires a two-step process: the empirical risk has to be minimized for each element of the structure. The optimal element $S^*$ is then selected to minimize the guaranteed risk, defined as the sum of the empirical risk and the confidence interval. This process involves a trade-off: as $h$ increases the minimum empirical risk decreases, but the confidence interval increases.

# 8   EXAMPLES OF STRUCTURES FOR NEURAL NETS

The general principle of SRM can be implemented in many different ways. Here we consider three different examples of structures built for the set of functions implemented by a neural network.

**1. Structure given by the architecture of the neural network.** Consider an ensemble of fully connected neural networks in which the number of units in one of the hidden layers is monotonically increased. The set of implementable functions makes a structure as the number of hidden units is increased.

**2. Structure given by the learning procedure.** Consider the set of functions $S = \{f(x, w), w \in W\}$ implementable by a neural net of fixed architecture. The parameters $\{w\}$ are the weights of the neural network. A structure is introduced through $S_p = \{f(x, w), \|w\| \leq C_p\}$ and $C_1 < C_2 < ... < C_n$. For a convex loss function, the minimization of the empirical risk within the element $S_p$ of the structure is achieved through the minimization of

$$E(w, \gamma_p) = \frac{1}{\ell} \sum_{i=1}^{\ell} L(y_i, f(x_i, w)) + \gamma_p \|w\|^2$$

with appropriately chosen Lagrange multipliers $\gamma_1 > \gamma_2 > ... > \gamma_n$. The well-known "weight decay" procedure refers to the minimization of this functional.

**3. Structure given by preprocessing.** Consider a neural net with fixed architecture. The input representation is modified by a transformation $z = K(x, \beta)$, where the parameter $\beta$ controls the degree of the degeneracy introduced by this transformation (for instance $\beta$ could be the width of a smoothing kernel).

A structure is introduced in the set of functions $S = \{f(K(x,\beta),w),\ w \in W\}$ through $\beta \geq C_p$, and $C_1 > C_2 > ... > C_n$.

## 9    PROBLEM OF LOCAL FUNCTION ESTIMATION

The problem of learning has been formulated as the problem of selecting from the class of functions $f(x,w),\ w \in W$ that which provides the best available approximation to the response of the supervisor. Such a statement of the learning problem implies that a unique function $f(x,w^*)$ will be used for prediction over the full input space $X$. This is not necessarily a good strategy: the set $f(x,w),\ w \in W$ might not contain a good predictor for the full input space, but might contain functions capable of good prediction on specified regions of input space.

In order to formulate the learning problem as a problem of local function approximation, consider a kernel $K(x - x_0, b) \geq 0$ which selects a region of input space of width $b$, centered at $x_0$. For example, consider the rectangular kernel,

$$K_r(x - x_0, b) = \left\{ \begin{array}{ll} 1 & \text{if } |x - x_0| \leq b \\ 0 & \text{otherwise} \end{array} \right.$$

and a more general general continuous kernel, such as the gaussian

$$K_g(x - x_0, b) = \exp -\{\frac{(x - x_0)^2}{b^2}\}.$$

The goal is to minimize the local risk functional

$$R(w, b, x_0) = \int L(y, f(x, w)) \frac{K(x - x_0, b)}{K(x_0, b)} dP(x, y). \tag{12}$$

The normalization is defined by

$$K(x_0, b) = \int K(x - x_0, b) \, dP(x). \tag{13}$$

The local risk functional (12) is to be minimized over the class of functions $f(x,w),\ w \in W$ and over all possible neighborhoods $b \in (0, \infty)$ centered at $x_0$. As before, the joint probability distribution $P(x, y)$ is unknown, and the only available information is contained in the training set (1).

## 10    EMPIRICAL RISK MINIMIZATION FOR LOCAL ESTIMATION

In order to solve this problem, the following induction principle is proposed: for fixed $b$, the local risk functional (12) is replaced by the empirical risk functional

$$E(w, b, x_0) = \frac{1}{\ell} \sum_{i=1}^{\ell} L(y_i, f(x_i, w)) \frac{K(x_i - x_0, b)}{K(x_0, b)}, \tag{14}$$

constructed on the basis of the training set. The empirical risk functional (14) is to be minimized over $w \in W$. In the simplest case, the class of functions is that of constant functions, $f(x, w) = C(w)$. Consider the following examples:

1. **K-Nearest Neighbors Method:** For the case of binary pattern recognition, the class of constant indicator functions contains only two functions: either $f(x, w) = 0$ for all $x$, or $f(x, w) = 1$ for all $x$. The minimization of the empirical risk functional (14) with the rectangular kernel $K_r(x - x_0, b)$ leads to the K-nearest neighbors algorithm.

2. **Watson-Nadaraya Method:** For the case $y \in R$, the class of constant functions contains an infinite number of elements, $f(x, w) = C(w)$, $C(w) \in R$. The minimization of the empirical risk functional (14) for general kernel and a quadratic loss function $L(y, f(x, w)) = (y - f(x, w))^2$ leads to the estimator

$$f(x_0) = \sum_{i=1}^{\ell} y_i \frac{K(x_i - x_0, b)}{\sum_{j=1}^{\ell} K(x_j - x_0, b)},$$

which defines the Watson-Nadaraya algorithm.

These classical methods minimize (14) with a fixed $b$ over the class of constant functions. The supervisor's response in the vicinity of $x_0$ is thus approximated by a constant, and the characteristic size $b$ of the neighborhood is kept fixed, independent of $x_0$.

A truly local algorithm would adjust the parameter $b$ to the characteristics of the region in input space centered at $x_0$. Further improvement is possible by allowing for a richer class of predictor functions $f(x, w)$ within the selected neighborhood. The SRM principle for local estimation provides a tool for incorporating these two features.

## 11  STRUCTURAL RISK MINIMIZATION FOR LOCAL ESTIMATION

The arguments that lead to the inequality (6) for the risk functional (2) can be extended to the local risk functional (12), to obtain the following result: with probability $1 - \eta$, and simultaneously for all $w \in W$ and all $b \in (0, \infty)$

$$R(w, b, x_0) < E(w, b, x_0) + C_2(\ell/h, b, \eta). \tag{15}$$

The confidence interval $C_2(\ell/h, b, \eta)$ reduces to $C_0(\ell/h, \eta)$ in the $b \to \infty$ limit.

As before, a nested structure is introduced in the class of functions, and the empirical risk (14) is minimized with respect to both $w \in W$ and $b \in (0, \infty)$ for each element of the structure. The optimal element is then selected to minimize the guaranteed risk, defined as the sum of the empirical risk and the confidence interval. For fixed $b$ this process involves an already discussed trade-off: as $h$ increases, the empirical risk decreases but the confidence interval increases. A new trade-off appears by varying $b$ at fixed $h$: as $b$ increases the empirical risk increases, but the confidence interval decreases. The use of $b$ as an additional free parameter allows us to find deeper minima of the guaranteed risk.

## 12  APPLICATION TO ZIP-CODE RECOGNITION

We now discuss results for the recognition of the hand written and printed digits in the US Postal database, containing 9709 training examples and 2007 testing examples. Human recognition of this task results in an approximately 2.5% prediction error (Säckinger et al., 1991).

The learning machine considered here is a five-layer neural network with shared weights and limited receptive fields. When trained with a back-propagation algorithm for the minimization of the empirical risk, the network achieves 5.1% prediction error (Le Cun et al., 1990).

Further performance improvement with the same network architecture has required the introduction a new induction principle. Methods based on SRM have achieved prediction errors of 4.1% (training based on a double-back-propagation algorithm which incorporates a special form of weight decay (Drucker, 1991)) and 3.95% (using a smoothing transformation in input space (Simard, 1991)).

The best result achieved so far, of 3.3% prediction error, is based on the use of the SRM for local estimation of the predictor function (Bottou, 1991).

It is obvious from these results that dramatic gains cannot be achieved through minor algorithmic modifications, but require the introduction of new principles.

### Acknowledgements

I thank the members of the Neural Networks research group at Bell Labs, Holmdel, for supportive and useful discussions. Sara Solla, Leon Bottou, and Larry Jackel provided invaluable help to render my presentation more clear and accessible to the neural networks community.

### References

V. N. Vapnik (1982), *Estimation of Dependencies Based on Empirical Data*, Springer-Verlag (New York).

V. N. Vapnik and A. Ja. Chervonenkis (1989) 'Necessary and sufficient conditions for consistency of the method of empirical risk minimization' [in Russian], *Yearbook of the Academy of Sciences of the USSR* on *Recognition, Classification, and Forecasting*, 2, 217-249, Nauka (Moscow) (English translation in preparation).

E. Säckinger and J. Bromley (1991), private communication.

Y. Le Cun, B. Boser, J. S. Denker, D. Henderson, R. E. Howard, W. Hubbard and L. D. Jackel (1990) 'Handwritten digit recognition with a back-propagation network', *Neural Information Processing Systems 2*, 396-404, ed. by D. S. Touretzky, Morgan Kaufmann (California).

H. Drucker (1991), private communication.

P. Simard (1991), private communication.

L. Bottou (1991), private communication.